# SpaRCS: Recovering Low-Rank and Sparse Matrices from Compressive Measurements

**Andrew E. Waters, Aswin C. Sankaranarayanan, Richard G. Baraniuk**
Rice University
{andrew.e.waters, saswin, richb}@rice.edu

## Abstract

We consider the problem of recovering a matrix $\mathbf{M}$ that is the sum of a low-rank matrix $\mathbf{L}$ and a sparse matrix $\mathbf{S}$ from a small set of linear measurements of the form $\mathbf{y} = \mathcal{A}(\mathbf{M}) = \mathcal{A}(\mathbf{L} + \mathbf{S})$. This model subsumes three important classes of signal recovery problems: compressive sensing, affine rank minimization, and robust principal component analysis. We propose a natural optimization problem for signal recovery under this model and develop a new greedy algorithm called SpaRCS to solve it. Empirically, SpaRCS inherits a number of desirable properties from the state-of-the-art CoSaMP and ADMiRA algorithms, including exponential convergence and efficient implementation. Simulation results with video compressive sensing, hyperspectral imaging, and robust matrix completion data sets demonstrate both the accuracy and efficacy of the algorithm.

## 1 Introduction

The explosion of digital sensing technology has unleashed a veritable data deluge that has pushed current signal processing algorithms to their limits. Not only are traditional sensing and processing algorithms increasingly overwhelmed by the sheer volume of sensor data, but storage and transmission of the data itself is also increasingly prohibitive without first employing costly compression techniques. This reality has driven much of the recent research on compressive data acquisition, in which data is acquired directly in a compressed format [1]. Recovery of the data typically requires finding a solution to an undetermined linear system, which becomes feasible when the underlying data possesses special structure. Within this general paradigm, three important problem classes have received significant recent attention: compressive sensing, affine rank minimization, and robust principal component analysis (PCA).

**Compressive sensing (CS):** CS is concerned with the recovery a vector $\mathbf{x}$ that is sparse in some transform domain [1]. Data measurements take the form $\mathbf{y} = \mathcal{A}(\mathbf{x})$, where $\mathcal{A}$ is an underdetermined linear operator. To recover $\mathbf{x}$, one would ideally solve

$$\min \ \|\mathbf{x}\|_0 \quad \text{subject to} \quad \mathbf{y} = \mathcal{A}(\mathbf{x}), \tag{1}$$

where $\|\mathbf{x}\|_0$ is the number of non-zero components in $\mathbf{x}$. This problem formulation is non-convex, CS recovery is typically accomplished either via convex relaxation or greedy approaches.

**Affine rank minimization:** The CS concept extends naturally to low-rank matrices. In the affine rank minimization problem [14, 23], we observe the linear measurements $\mathbf{y} = \mathcal{A}(\mathbf{L})$, where $\mathbf{L}$ is a low-rank matrix. One important sub-problem is that of matrix completion [3, 5, 22], where $\mathcal{A}$ takes the form of a sampling operator. To recover $\mathbf{L}$, one would ideally solve

$$\min \ \text{rank}(\mathbf{L}) \quad \text{subject to} \quad \mathbf{y} = \mathcal{A}(\mathbf{L}). \tag{2}$$

As with CS, this problem is non-convex and so several algorithms based on convex relaxation and greedy methods have been developed for finding solutions.

**Robust PCA:** In the robust PCA problem [2, 8], we wish to decompose a matrix $\mathbf{M}$ into a low-rank matrix $\mathbf{L}$ and a sparse matrix $\mathbf{S}$ such that $\mathbf{M} = \mathbf{L} + \mathbf{S}$. This problem is known to have a stable solution provided $\mathbf{L}$ and $\mathbf{S}$ are sufficiently incoherent [2]. To date, this problem has been studied only in the non-compressive setting, i.e, when $\mathbf{M}$ is fully available. A variety of convex relaxation methods have been proposed for solving this case.

The work of this paper stands at the intersection of these three problems. Specifically, we aim to recover the entries of a matrix $\mathbf{M}$ in terms of a low-rank matrix $\mathbf{L}$ and sparse matrix $\mathbf{S}$ *from a small set of compressive measurements* $\mathbf{y} = \mathcal{A}(\mathbf{L} + \mathbf{S})$. This problem is relevant in several application settings. A first application is the recovery of a video sequence obtained from a static camera observing a dynamic scene under changing illumination. Here, each column of $\mathbf{M}$ corresponds to a vectorized image frame of the video. The changing illumination has low-rank properties, while the foreground innovations exhibit sparse structures [2]. In such a scenario, neither sparse nor low-rank models are individually sufficient for capturing the underlying information of the signal. Models that combine low-rank and sparse components, however, are well suited for capturing such phenomenon. A second application is hyperspectral imaging, where each column of $\mathbf{M}$ is the vectorized image of a particular spectral band; a low-rank plus sparse model arises naturally due to material properties [7]. A third application is robust matrix completion [11], which can be cast as a compressive low-rank and sparse recovery problem.

The natural optimization problem that unites the above three problem classes above is

$$(\text{P1}) \quad \min \ \|\mathbf{y} - \mathcal{A}(\mathbf{L} + \mathbf{S})\|_2 \quad \text{subject to} \quad \text{rank}(\mathbf{L}) \leq r, \ \|\text{vec}(\mathbf{S})\|_0 \leq K. \quad (3)$$

The main contribution of this paper is a novel greedy algorithm for solving (P1), which we dub *SpaRCS* for SPArse and low Rank decomposition via Compressive Sensing. To the best of our knowledge, we are the first to propose a computationally efficient algorithm for solving a problem like (P1). SpaRCS combines the best aspects of CoSaMP [20] for sparse vector recovery and ADMiRA [17] for low-rank matrix recovery.

## 2 Background

Here we introduce the relevant background information regarding signal recovery from CS measurements, where our definition of signal is broadened to include both vectors and matrices. We further provide background on incoherency between low-rank and sparse matrices.

**Restricted isometry and rank-restricted isometry properties:** Signal recovery for a $K$-sparse vector from CS measurements is possible when the measurement operator $\mathcal{A}$ obeys the so-called restricted isometry property (RIP) [4] with constant $\delta_K$

$$(1 - \delta_K)\|\mathbf{x}\|_2^2 \leq \|\mathcal{A}(\mathbf{x})\|_2^2 \leq (1 + \delta_K)\|\mathbf{x}\|_2^2, \quad \forall \|\mathbf{x}\|_0 \leq K. \quad (4)$$

This property implies that the information in $\mathbf{x}$ is nearly preserved after being measured by $\mathcal{A}$. Analogous to CS, it has been shown that a low-rank matrix can be recovered from a set of CS measurements when the measurement operator $\mathcal{A}$ obeys the rank-restricted isometry (RRIP) property [23] with constant $\delta_r^*$

$$(1 - \delta_r^*)\|\mathbf{L}\|_F^2 \leq \|\mathcal{A}(\mathbf{L})\|_F^2 \leq (1 + \delta_r^*)\|\mathbf{L}\|_F^2, \quad \forall \text{rank}(\mathbf{L}) \leq r. \quad (5)$$

**Recovery algorithms:** Recovery of sparse vectors and low-rank matrices can be accomplished when the measurement operator $\mathcal{A}$ satisfies the appropriate RIP or RRIP condition. Recovery algorithms typically fall into one of two broad classes: convex optimization and greedy iteration. Convex optimization techniques recast (1) or (2) in a form that can be solved efficiently using convex programming [2, 27]. In the case of CS, the $\ell_0$ norm is relaxed to the $\ell_1$ norm; for low-rank matrices, the rank operator is relaxed to the nuclear norm.

In contrast, greedy algorithms [17, 20] operate iteratively on the signal measurements, constructing a basis for the signal and attempting signal recovery restricted to that basis. Compared to convex approaches, these algorithms often have superior speed and scale better to large problems. We highlight the CoSaMP algorithm [20] for sparse vector recovery and the ADMiRA algorithm [17] for low-rank matrix recovery in this paper. Both algorithms have strong convergence guarantees when the measurement operator $\mathcal{A}$ satisfies the appropriate RIP or RRIP condition, most notably exponential convergence to the true signal.

**Matrix Incoherency:** For matrix decomposition problems such as the Robust PCA problem or the problem defined in (3) to have unique solutions, there must exist a degree of incoherence between the low-rank matrix $\mathbf{L}$ and the sparse matrix $\mathbf{S}$. It is known that the decomposition of a matrix into its low-rank and sparse components makes sense only when the low-rank matrix is not sparse and, similarly, when the sparse matrix is not low-rank. A simple deterministic condition can be found in the work by Chandrasekaran, et al [9].

For our purposes, we assume the following model for non-sparse low rank matrices.

**Definition 2.1 (Uniformly bounded matrix [5])** *An* $N \times N$ *matrix* $\mathbf{L}$ *of rank* $r$ *is uniformly bounded if its singular vectors* $\{\mathbf{u}_j, \mathbf{v}_j, 1 \leq j \leq r\}$ *obey*

$$\|\mathbf{u}_j\|_\infty, \|\mathbf{v}_j\|_\infty \leq \sqrt{\mu_B/N},$$

*with* $\mu_B = O(1)$*, where* $\|\mathbf{x}\|_\infty$ *denotes the largest entry in magnitude of* $\mathbf{x}$*.*

When $\mu_B$ is small (note that $\mu_B \geq 1$), this model for the low-rank matrix $\mathbf{L}$ ensures that its singular vectors are not sparse. This can be seen in the case of the a singular vector $\mathbf{u}$ by noting that $1 = \|\mathbf{u}\|_2^2 = \sum_{k=1}^N u_k^2 \leq \|\mathbf{u}\|_0 \|\mathbf{u}\|_\infty^2$. Rearranging terms enables us to write $\|\mathbf{u}\|_0 \geq \frac{1}{\|\mathbf{u}\|_\infty^2} \geq \frac{N}{\mu_B}$. Thus, $\mu_B$ controls the sparsity of the matrix $\mathbf{L}$ by bounding the sparsity of its singular vectors.

A sufficient model for a sparse matrix that is not low-rank is to assume that the support set $\Omega$ is uniform. As shown in the work of Candes, et al [2] this model is equivalent to defining the sparse support set $\Omega = \{(i,j) : \delta_{i,j} = 1\}$ with each $\delta_{i,j}$ being an i.i.d. Bernoulli with sufficiently small parameter $\rho_S$.

## 3   SpaRCS: CS recovery of low-rank and sparse matrices

We now present the SpaRCS algorithm to solve (P1) and disucss its empirical properties. Assume that we are interested in a matrix $\mathbf{M} \in \mathbb{R}^{N_1 \times N_2}$ such that $\mathbf{M} = \mathbf{L} + \mathbf{S}$, with $\mathrm{rank}(\mathbf{L}) \leq r$, $\mathbf{L}$ uniformly bounded with constant $\mu_B$, and $\|\mathbf{S}\|_0 \leq K$ with support distributed uniformly. Further assume that a known linear operator $\mathcal{A} : \mathbb{R}^{N_1 \times N_2} \to \mathbb{R}^p$ provides us with $p$ compressive measurements $\mathbf{y}$ of $\mathbf{M}$. Let $\mathcal{A}^*$ denote the adjoint of the operator $\mathcal{A}$ and, given the index set $T \subset \{1, \ldots, N_1 N_2\}$, let $\mathcal{A}_{|T}$ denote the restriction of the operator to $T$. Given $\mathbf{y} = \mathcal{A}(\mathbf{M}) + \mathbf{e}$, where $\mathbf{e}$ denotes measurement noise, our goal is to estimate a low rank matrix $\widehat{\mathbf{L}}$ and a sparse matrix $\widehat{\mathbf{S}}$ such that $\mathbf{y} \approx \mathcal{A}(\widehat{\mathbf{L}} + \widehat{\mathbf{S}})$.

### 3.1   Algorithm

SpaRCS iteratively estimates $\mathbf{L}$ and $\mathbf{S}$; the estimation of $\mathbf{L}$ is closely related to ADMiRA [17], while the estimation of $\mathbf{S}$ is closely related to CoSaMP [20]. At each iteration, SpaRCS computes a signal proxy and then proceeds through four steps to update its estimates of $\mathbf{L}$ and $\mathbf{S}$. These steps are laid out in Algorithm 1. We use the notation $\mathrm{supp}(\mathbf{X}; K)$ to denote the largest $K$-term support set of the matrix $\mathbf{X}$. This forms a natural basis for sparse signal approximation. We further use the notation $\mathrm{svd}(\mathbf{X}; r)$ to denote computation of the $\mathrm{rank}$-$r$ singular value decomposition (SVD) of $\mathbf{X}$ and the arrangement of its singular vectors into a set of up to $r$ rank-1 matrices. This set of rank-1 matrices serve as a natural basis for approximating uniformly bounded low-rank matrices.

### 3.2   Performance characterization

Empirically, SpaRCS produces a series of estimates $\widehat{\mathbf{L}}_k$ and $\widehat{\mathbf{S}}_k$ that converge convergence exponentially towards the true values $\mathbf{L}$ and $\mathbf{S}$. This performance is inhereted largely from the behavior of the CoSaMP and ADMiRA algorithms with one noteworthy modification. The key difference is that, for SpaRCS, the sparse and low-rank estimation problems are coupled. While CoSaMP and ADMiRA operate solely in the presence of the measurement noise, SpaRCS must estimate $\mathbf{L}$ in the presence of the residual error of $\mathbf{S}$, and vice-versa. Proving convergence for the algorithm in the presence of the additional residual terms is non-trivial; simply lumping these additional residual errors together with the measurement noise $\mathbf{e}$ is insufficient for analysis.

As a concrete example, consider the support identification step $\widehat{\mathbf{\Psi}}_{\mathbf{S}} \leftarrow \mathrm{supp}(\mathbf{P}; 2K)$, with

$$\mathbf{P} = \mathcal{A}^*(\mathbf{w}_{k-1}) = \mathcal{A}^*(\mathcal{A}(\mathbf{S} - \widehat{\mathbf{S}}_{k-1}) + \mathcal{A}(\mathbf{L} - \widehat{\mathbf{L}}_{k-1}) + \mathbf{e}),$$

**Algorithm 1:** $(\widehat{\mathbf{L}}, \widehat{\mathbf{S}}) = \text{SpaRCS} \, (\mathbf{y}, \mathcal{A}, \mathcal{A}^*, K, r, \epsilon)$

---

Initialization: $k \leftarrow 1, \widehat{\mathbf{L}}_0 \leftarrow \mathbf{0}, \widehat{\mathbf{S}}_0 \leftarrow \mathbf{0}, \mathbf{\Psi_L} \leftarrow \emptyset, \mathbf{\Psi_S} \leftarrow \emptyset, \mathbf{w}_0 \leftarrow \mathbf{y}$

**while** $\|\mathbf{w}_{k-1}\|_2 \geq \epsilon$ **do**

> Compute signal proxy:
> $\quad \mathbf{P} \leftarrow \mathcal{A}^*(\mathbf{w}_{k-1})$
> Support identification:
> $\quad \widehat{\mathbf{\Psi}}_{\mathbf{L}} \leftarrow \text{svd}(\mathbf{P}; 2r); \; \widehat{\mathbf{\Psi}}_{\mathbf{S}} \leftarrow \text{supp}(\mathbf{P}; 2K)$
> Support merger:
> $\quad \widetilde{\mathbf{\Psi}}_{\mathbf{L}} \leftarrow \widehat{\mathbf{\Psi}}_{\mathbf{L}} \bigcup \mathbf{\Psi_L}; \; \widetilde{\mathbf{\Psi}}_{\mathbf{S}} \leftarrow \widehat{\mathbf{\Psi}}_{\mathbf{S}} \bigcup \mathbf{\Psi_S}$
> Least squares estimation:
> $\quad \mathbf{B}^{\mathbf{L}} \leftarrow \widetilde{\mathbf{\Psi}}_{\mathbf{L}}^{\dagger}(\mathbf{y} - \mathcal{A}(\widehat{\mathbf{S}}_{k-1})); \mathbf{B}^{\mathbf{S}} \leftarrow \widetilde{\mathbf{\Psi}}_{\mathbf{S}}^{\dagger}(\mathbf{y} - \mathcal{A}(\widehat{\mathbf{L}}_{k-1}))$
> Support pruning:
> $\quad (\widehat{\mathbf{L}}_k \, , \, \mathbf{\Psi_L}) \leftarrow \text{svd}(\mathbf{B}^{\mathbf{L}}; r); \; (\widehat{\mathbf{S}}_k \, , \, \mathbf{\Psi_S}) \leftarrow \text{supp}(\mathbf{B}^{\mathbf{S}}; K)$
> Update residue:
> $\quad \mathbf{w}_k \leftarrow \mathbf{y} - \mathcal{A}(\widehat{\mathbf{L}}_k + \widehat{\mathbf{S}}_k)$
>
> $k \leftarrow k + 1$

**end**

$\widehat{\mathbf{L}} = \widehat{\mathbf{L}}_{k-1}; \widehat{\mathbf{S}} = \widehat{\mathbf{S}}_{k-1}$

---

that estimates the support set of $\mathbf{S}$. CoSaMP relies on high correlation between $\text{supp}(\mathbf{P}; 2K)$ and $\text{supp}(\mathbf{S} - \widehat{\mathbf{S}}_{k-1}; 2K)$; to achieve the same in SpaRCS, $(\mathbf{L} - \widehat{\mathbf{L}}_{k-1})$ must be well behaved.

We are currently preparing a full theoretical characterization of the SpaRCS algorithm along with the necessary conditions that guarantee this exponential convergence property. We reserve the presentation of the convergence proof for an extended version of this work.

**Phase transition:** The empirical performance of SpaRCS can be charted using phase transition plots, which predicts sufficient and necessary conditions on its success/failure. Figure 1 shows phase transition results on a problem of size $N_1 = N_2 = 512$ for various values of $p$, $r$, and $K$. As expected, SpaRCS degrades gracefully as we decrease $p$ or increase $r$ and $K$.

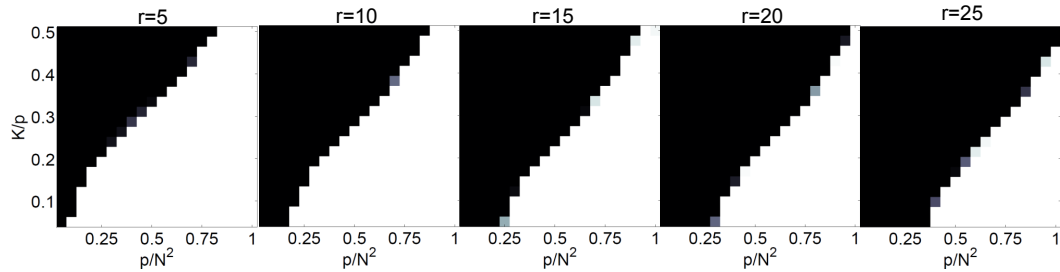

Figure 1: Phase transitions for a recovery problem of size $N_1 = N_2 = N = 512$. Shown are aggregate results over 20 Monte-Carlo runs at each specification of $r, K$, and $p$. Black indicates recovery failure, while white indicates recovery success.

**Computational cost:** SpaRCS is highly computationally efficient and scales well as $N_1, N_2$ grow large. The largest computational cost is that of computing the two truncated SVDs per iteration. The SVDs can be performed efficiently via the Lanczos algorithm or similar method. The least squares estimation can be solved efficiently using conjugate gradient or Richardson iterations. Support estimation for the sparse vector merely entails sorting the signal proxy magnitudes and choosing the largest $2K$ elements.

Figure 2 compares the performance of SpaRCS with two alternate recovery algorithms. We implement CS versions of the IT [18] and APG [19] algorithms, which solve the problems

$$\min \tau \left( \|\mathbf{L}\|_* + \lambda \|\mathrm{vec}(\mathbf{S})\|_1 \right) + \frac{1}{2}\|\mathbf{L}\|_F^2 + \frac{1}{2}\|\mathbf{S}\|_F^2 \text{ s.t. } \mathbf{y} = \mathcal{A}(\mathbf{L} + \mathbf{S})$$

and

$$\min \|\mathbf{L}\|_* + \lambda \|\mathrm{vec}(\mathbf{S})\|_1 \text{ s.t. } \mathbf{y} = \mathcal{A}(\mathbf{L} + \mathbf{S}),$$

respectively. We endeavor to tune the parameters of these algorithms (which we refer to as CS IT and CS APG, respectively) to optimize their performance. Details of our implementation can be found in [26]. In all experiments, we consider matrices of size $N \times N$ with $\mathrm{rank}(\mathbf{L}) = 2$ and $\|\mathbf{S}\|_0 = 0.02N^2$ and use permuted noiselets [12] for the measurement operator $\mathcal{A}$. As a first experiment, we generate convergence plots for matrices with $N = 128$ and vary the measurement ratio $p/N^2$ from 0.05 to 0.5. We then recover $\widehat{\mathbf{L}}$ and $\widehat{\mathbf{S}}$ and measure the recovered signal to noise ratio (RSNR) for $\widehat{\mathbf{M}} = \widehat{\mathbf{L}} + \widehat{\mathbf{S}}$ via $20 \log_{10}\left(\frac{\|\mathbf{M}\|_F}{\|\mathbf{M}-\widehat{\mathbf{L}}-\widehat{\mathbf{S}}\|_F}\right)$. These results are displayed in Figure 2(a), where we see that SpaRCS provides the best recovery. As a second experiment, we vary the problem size $N \in \{128, 256, 512, 1024\}$ while holding the number of measurements constant at $p = 0.2N^2$. We measure the recovery time required by each algorithm to reach a residual error $\frac{\|y-A(\widehat{\mathbf{L}}+\widehat{\mathbf{S}})\|_2}{\|\mathbf{y}\|_2} \leq 5 \times 10^{-4}$. These results are displayed in Figure 2(b), which demonstrate that SpaRCS converges significantly faster than the two other recovery methods.

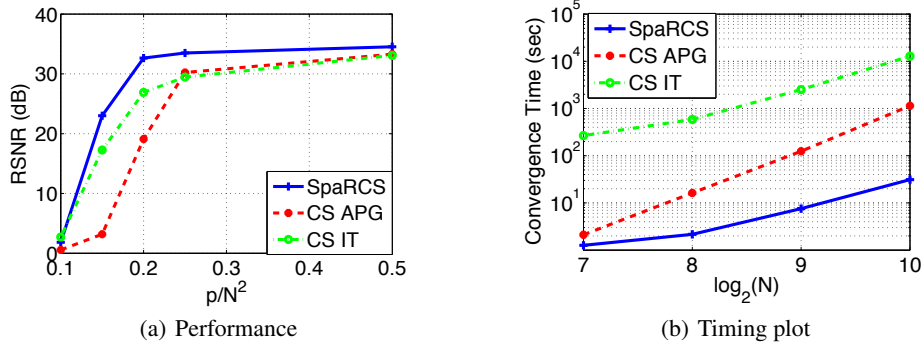

(a) Performance                    (b) Timing plot

Figure 2: Performance and run-time comparisons between SpaRCS, CS IT, and CS APG. Shown are average results over 10 Monte-Carlo runs for problems of size $N_1 = N_2 = N$ with $\mathrm{rank}(\mathbf{L}) = 2$ and $\|\mathbf{S}\|_0 = 0.02N^2$. (a) Performance for a problem with $N = 128$ for various values of the measurement ratio $p/N^2$. SpaRCS exhibits superior recovery over the alternate approaches. (b) Timing plot for problems of various sizes $N$. SpaRCS converges in time several orders of magnitude faster than the alternate approaches.

## 4   Applications

We now present several experiments that validate SpaRCS and showcase its performance in several applications. In all experiments, we use permuted noiselets for the measurement operator $\mathcal{A}$; these provide both a fast transform as well as save memory, since we do not have to store $\mathcal{A}$ explicitly.

**Video compressive sensing:** The video CS problem is concerned with recovering multiple image frames of a video sequence from CS measurements [6, 21, 24]. We consider a $128 \times 128 \times 201$ video sequence consisting of a static background with a number of people moving in the foreground. We aim to not only recover the original video but also separate the background and foreground. We resize the data cube into a $128^2 \times 201$ matrix $\mathbf{M}$, where each column corresponds to a (vectorized) image frame. The measurement operator $\mathcal{A}$ operates on each column of $\mathbf{M}$ independently, simulating acquisition using a single pixel camera [13]. We acquire $p = 0.15 \times 128^2$ measurements per image frame. We recover with SpaRCS using $r = 1$ and $K = 20{,}000$. The results are displayed in Figure 3, where it can be seen that SpaRCS accurately estimates and separates the low-rank background and the sparse foreground. Figure 4 shows recovery results on a more challenging sequence

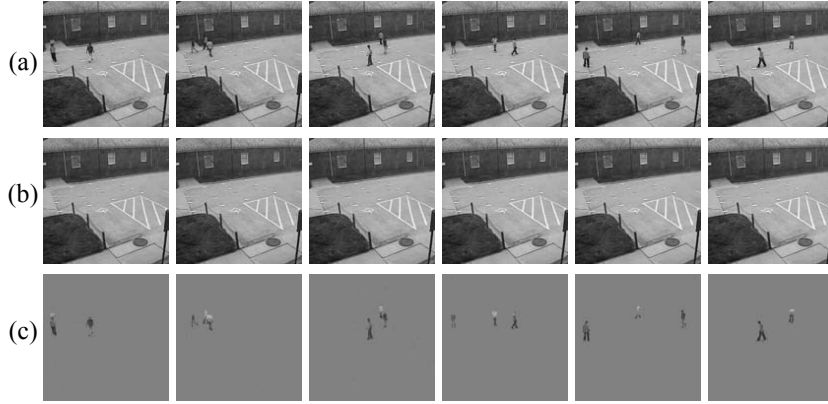

Figure 3: SpaRCS recovery results on a $128 \times 128 \times 201$ video sequence. The video sequence is reshaped into an $N_1 \times N_2$ matrix with $N_1 = 128^2$ and $N_2 = 201$. (a) Ground truth for several frames. (b) Estimated low-rank component **L**. (c) Estimated sparse component **S**. The recovery SNR is 31.2 dB at the measurement ratio $p/(N_1 N_2) = 0.15$. The recovery is accurate in spite of the measurement operator $\mathcal{A}$ working independently on each frame.

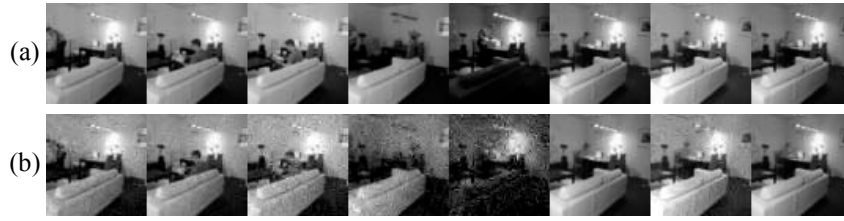

Figure 4: SpaRCS recovery results on a $64 \times 64 \times 234$ video sequence. The video sequence is reshaped into an $N_1 \times N_2$ matrix with $N_1 = 64^2$ and $N_2 = 234$. (a) Ground truth for several frames. (b) Recovered frames. The recovery SNR is 23.9 dB at the measurement ratio of $p/(N_1 N_2) = 0.33$. The recovery is accurate in spite of the changing illumination conditions.

with changing illumination. In contrast to SpaRCS, existing video CS algorithms do not work well with dramatically changing illumination.

**Hyperspectral compressive sensing:** Low-rank/sparse decomposition has an important physical relevance in hyperspectral imaging [7]. Here we consider a hyperspectral cube, which contains a vector of spectral information at each image pixel. A measurement device such as [25] can provide compressive measurements of such a hyperspectral cube. We employ SpaRCS on a hyperspectral cube of size $128 \times 128 \times 128$ rearranged as a matrix of size $128^2 \times 128$ such that each column corresponds to a different spectral band. Figure 5 demonstrates recovery using $p = 0.15 \times 128^2 \times 128$ total measurements of the entire data cube with $r = 8$, $K = 3000$. SpaRCS performs well in terms of residual error (Figure 5(c)) despite the number of rows being much larger than the number of columns. Figure 5(d) emphasizes the utility the sparse component. Using only a low-rank approximation (corresponding to traditional PCA) causes a significant increase in residual error over what is achieved by SpaRCS.

**Parameter mismatch:** In Figure 6, we analyze the influence of incorrect selection of the parameters $r$ using the hyperspectral data as an example. We plot the recovered SNR that can be obtained at various levels of the measurement ratio $p/(N_1 N_2)$ for both the case of $r = 8$ and $r = 4$. There are interesting tradeoffs associated with the choice of parameters. Larger values of $r$ and $K$ enable better approximation to the unknown signals. However, by increasing $r$ and $K$, we also increase the number of independent parameters in the problem, which is given by $(2 \max(N_1, N_2)r - r^2 + 2K)$. An empirical rule-of-thumb for greedy recovery algorithms is that the number of measurements $p$ should be 2–5 times the number of independent parameters. Consequently, there exists a tradeoff between the values of $r$, $K$, and $p$ to ensure stable recovery.

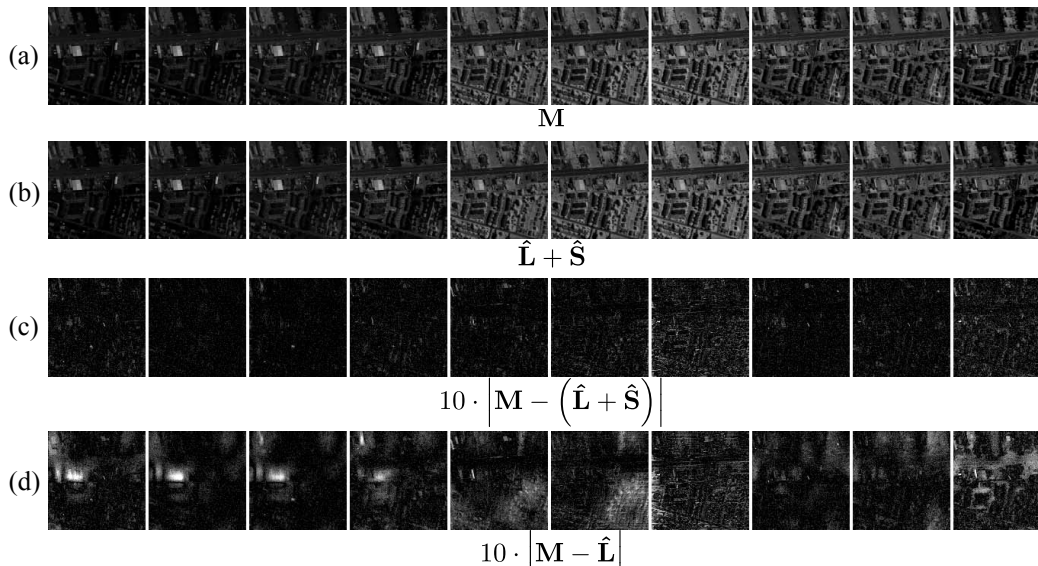

Figure 5: SpaRCS recovery results on a $128 \times 128 \times 128$ hyperspectral data cube. The hyperspectral data is reshaped into an $N_1 \times N_2$ matrix with $N_1 = 128^2$ and $N_2 = 128$. Each image pane corresponds to a different spectral band. (a) Ground truth. (b) Recovered images. (c) Residual error using both the low-rank and sparse component. (d) Residual error using only the low-rank component. The measurement ratio is $p/(N_1 N_2) = 0.15$.

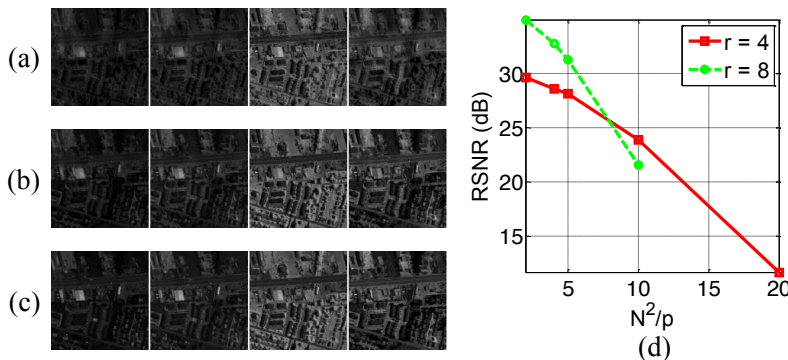

Figure 6: Hyperspectral data recovery for various values of the rank $r$ of the low-rank matrix $\mathbf{L}$. The data used is the same as in Figure 5. (a) $r = 1$, SNR = 12.81 dB. (b) $r = 2$, SNR = 19.42 dB. (c) $r = 4$, SNR = 27.46 dB. (d) Comparison of compression ratio $(N_1 N_2)/p$ and recovery SNR using $r = 4$ and $r = 8$. All results were obtained with $K = 3000$.

**Robust matrix completion:** We apply SpaRCS to the robust matrix completion problem [11]

$$\min \|\mathbf{L}\|_* + \lambda \|\mathbf{s}\|_1 \quad \text{subject to} \quad \mathbf{L}_\Omega + \mathbf{s} = \mathbf{y} \tag{6}$$

where $\mathbf{s}$ models outlier noise and $\Omega$ denotes the set of observed entries. This problem can be cast as a compressive low-rank and sparse matrix recovery problem by using a sparse matrix $\mathbf{S}$ in place of the outlier noise $\mathbf{s}$ and realizing that the support of $\mathbf{S}$ is a subset of $\Omega$. This enables recovery of both $\mathbf{L}$ and $\mathbf{S}$ from samples of their sum $\mathbf{L} + \mathbf{S}$.

Matrix completion under outlier noise [10, 11] has received some attention and, in many ways, is the work that is closest to this paper. There are, however, several important distinctions. Chen et al. [11] analyze the convex problem of (6) to provide performance guarantees. Yet, convex optimization methods often do not scale well with the size of the problem. SpaRCS, by contrast, is computationally efficient and does scale well as the problem size increases. Furthermore, [10] is tied to the case when $\mathcal{A}$ is a sampling operator; it is not immediately clear whether this analysis can extend to the more general case of (P1), where the sparse component cannot be modeled as outlier noise in the measurements.

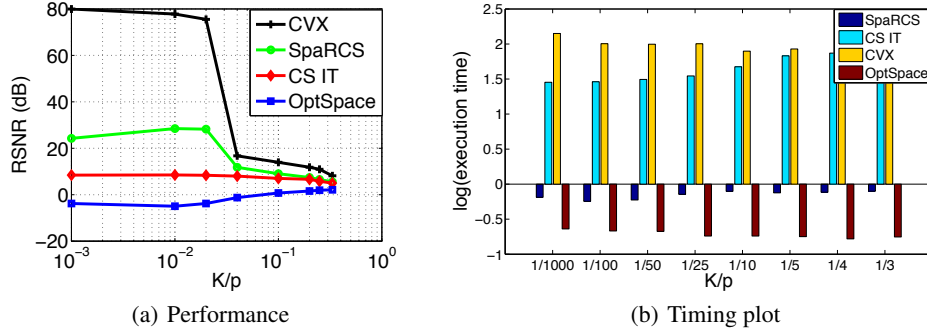

| (a) Performance | (b) Timing plot |

Figure 7: Comparison of several algorithms for the robust matrix completion problem. (a) RSNR averaged over 10 Monte-Carlo runs for an $N \times N$ matrix completion problem with $N = 128$, $r = 1$, and $p/N^2 = 0.2$. Non-robust formulations, such OptSpace, fail. SpaRCS acheives performance close to that of the convex solver (CVX). (b) Comparison of convergence times for the various algorithms. SpaRCS converges in only a fraction of the time required by the other algorithms.

In our robust matrix completion experiments we compare SpaRCS with CS SVT, OptSpace [16] (a non-robust matrix completion algorithm), and a convex solution using CVX [15]. Figure 7 shows the performance of these algorithms. OptSpace, being non-robust, fails as expected. The accuracy of SpaRCS is closest to that of CVX, although the convergence time of SpaRCS is several orders of magnitude faster.

## 5    Conclusion

We have considered the problem of recovering low-rank and sparse matrices given only a few linear measurements. Our proposed greedy algorithm, SpaRCS, is both fast and accurate even for large matrix sizes and enjoys strong empirical performance in its convergence to the true solution. We have demonstrated the applicability of SpaRCS to video compressive sensing, hyperspectral imaging, and robust matrix completion.

There are many avenues for future work. Model-based extensions of SpaRCS are important directions. Both low-rank and sparse matrices exhibit rich structure in practice, including low-rank Hankel matrices in system identification and group sparsity in background subtraction. The use of models could significantly enhance the performance of the algorithm. This would be especially useful in applications such as video CS, where the measurement operator is typically constrained to operate on each image frame individually.

## Acknowledgements

This work was partially supported by the grants NSF CCF-0431150, CCF-0728867, CCF-0926127, CCF-1117939, ARO MURI W911NF-09-1-0383, W911NF-07-1-0185, DARPA N66001-11-1-4090, N66001-11-C-4092, N66001-08-1-2065, AFOSR FA9550-09-1-0432, and LLNL B593154.

Additionally, the authors wish to thank Prof. John Wright for his helpful comments and corrections to a previous version of this manuscript.

# References

[1] E. J. Candès. Compressive sampling. In *Intl. Cong. of Math.*, Madrid, Spain, Aug. 2006.

[2] E. J. Candès, X. Li, Y. Ma, and J. Wright. Robust principal component analysis? *J. ACM*, 58(1):1–37, 2009.

[3] E. J. Candès and Y. Plan. Matrix completion with noise. *Proc. IEEE*, 98(6):925–936, 2010.

[4] E. J. Candès and J. Romberg. Quantitative robust uncertainty principles and optimally sparse decompositions. *Found. Comput. Math.*, 6(2):227–254, 2006.

[5] E.J. Candès and T. Tao. The power of convex relaxation: Near-optimal matrix completion. *IEEE Trans. on Info. Theory*, 56(5):2053–2080, 2010.

[6] V. Cevher, A. C. Sankaranarayanan, M. Duarte, D. Reddy, R. G. Baraniuk, and R. Chellappa. Compressive sensing for background subtraction. In *European Conf. Comp. Vision*, Marseilles, France, Oct. 2008.

[7] A. Chakrabarti and T. Zickler. Statistics of Real-World Hyperspectral Images. In *IEEE Int. Conf. Comp. Vis.*, Colorado Springs, CO, June 2011.

[8] V. Chandrasekaran, S. Sanghavi, P. A. Parrilo, and A. S. Willsky. Sparse and low-rank matrix decompositions. In *Allerton Conf. on Comm., Contr., and Comp.*, Monticello, IL, Sep. 2009.

[9] V. Chandrasekaran, S. Sanghavi, P. A. Parrilo, and A.S. Willsky. Rank-sparsity incoherence for matrix decomposition. *Arxiv preprint arXiv:0906.2220*, 2009.

[10] Y. Chen, A. Jalali, S. Sanghavi, and C. Caramanis. Low-rank matrix recovery from errors and erasures. *Arxiv preprint arXiv:1104.0354*, 2011.

[11] Y. Chen, H. Xu, C. Caramanis, and S. Sanghavi. Robust matrix completion with corrupted columns. *Arxiv preprint arXiv:1102.2254*, 2011.

[12] R. Coifman, F. Geshwind, and Y. Meyer. Noiselets. *Appl. Comput. Harmon. Anal.*, 10:27–44, 2001.

[13] M. F. Duarte, M. A. Davenport, D. Takhar, J. N. Laska, T. Sun, K. F. Kelly, and R. G. Baraniuk. Single pixel imaging via compressive sampling. *IEEE Signal Processing Mag.*, 25(2):83–91, 2008.

[14] M. Fazel, E. Candès, B. Recht, and P. Parrilo. Compressed sensing and robust recovery of low rank matrices. In *Asilomar Conf. Signals, Systems, and Computers*, Pacific Grove, CA, Nov. 2008.

[15] M. Grant and S. Boyd. CVX: Matlab software for disciplined convex programming, version 1.21. http://cvxr.com/cvx, Apr. 2011.

[16] R. H. Keshavan, A. Montanari, and S. Oh. Matrix completion from noisy entries. *J. Mach. Learn. Res.*, 11:2057–2078, 2010.

[17] K. Lee and Y. Bresler. Admira: Atomic decomposition for minimum rank approximation. *IEEE Trans. on Info. Theory*, 56(9):4402–4416, 2010.

[18] Z. Lin, M. Chen, L. Wu, and Y. Ma. The augmented lagrange multiplier method for exact recovery of corrupted low-rank matrices. Technical report, University of Illinois at Urbana-Champaign, Urbana-Champaign, IL.

[19] Z. Lin, A. Ganesh, J. Wright, L. Wu, M. Chen, and Y. Ma. Fast convex optimization algorithms for exact recovery of a corrupted low-rank matrix. In *Intl. Workshop on Comp. Adv. in Multi-Sensor Adapt. Processing*, Aruba, Dutch Antilles, Dec. 2009.

[20] D. Needell and J.A. Tropp. Cosamp: Iterative signal recovery from incomplete and inaccurate samples. *Appl. Comput. Harmon. Anal.*, 26(3):301–321, 2009.

[21] J. Y. Park and M. B. Wakin. A multiscale framework for compressive sensing of video. In *Picture Coding Symp.*, Chicago, IL, May 2009.

[22] B. Recht. A simpler approach to matrix completion. *J. Mach. Learn. Res.*, posted Oct. 2009, to appear.

[23] B. Recht, M. Fazel, and P. A. Parrilo. Guaranteed minimum rank solutions of linear matrix equations via nuclear norm minimization. *SIAM Rev.*, 52(3):471–501, 2010.

[24] A. C. Sankaranarayanan, P. Turaga, R. G. Baraniuk, and R. Chellappa. Compressive acquisition of dynamic scenes. In *European Conf. Comp. Vision*, Crete, Greece, Sep. 2010.

[25] T. Sun and K. Kelly. Compressive sensing hyperspectral imager. In *Comput. Opt. Sensing and Imaging*, San Jose, CA, Oct. 2009.

[26] A. E. Waters, A. C. Sankaranarayanan, and R. G. Baraniuk. SpaRCS: Recovering low-rank and sparse matrices from compressive measurements. Technical report, Rice University, Houston, TX, 2011.

[27] W. Yin, S. Osher, D. Goldfarb, and J. Darbon. Bregman iterative algorithms for $\ell_1$-minimization with applications to compressed sensing. *SIAM J. Imag. Sci.*, 1(1):143–168, 2008.

